# Co-Validation: Using Model Disagreement on Unlabeled Data to Validate Classification Algorithms

**Omid Madani, David M. Pennock, Gary W. Flake**
Yahoo! Research Labs
3rd floor, Pasadena Ave.
Pasadena, CA 91103
{madani|pennockd|flakeg}@yahoo-inc.com

## Abstract

In the context of binary classification, we define *disagreement* as a measure of how often two independently-trained models differ in their classification of unlabeled data. We explore the use of disagreement for error estimation and model selection. We call the procedure *co-validation*, since the two models effectively (in)validate one another by comparing results on unlabeled data, which we assume is relatively cheap and plentiful compared to labeled data. We show that per-instance disagreement is an unbiased estimate of the variance of error for that instance. We also show that disagreement provides a lower bound on the prediction (generalization) error, and a tight upper bound on the "variance of prediction error", or the variance of the average error across instances, where variance is measured across training sets. We present experimental results on several data sets exploring co-validation for error estimation and model selection. The procedure is especially effective in active learning settings, where training sets are not drawn at random and cross validation overestimates error.

## 1   Introduction

Balancing hypothesis-space generality with predictive power is one of the central tasks in inductive learning. The difficulties that arise in seeking an appropriate tradeoff go by a variety of names—overfitting, data snooping, memorization, no free lunch, bias-variance tradeoff, etc.—and lead to a number of known solution techniques or philosophies, including regularization, minimum description length, model complexity penalization (e.g., BIC, AIC), Ockham's razor, training with noise, ensemble methods (e.g., boosting), structural risk minimization (e.g., SVMs), cross validation, hold-out validation, etc.

All of these methods in some way attempt to estimate or control the prediction (generalization) error of an induced function on unseen data. In this paper, we explore a method of error estimation that we call *co-validation*. The method trains two independent functions that in a sense validate (or invalidate) one another by examining their mutual rate of *disagreement* across a set of unlabeled data. In Section 2, we formally define disagreement. The measure simultaneously reflects notions of algorithm stability, model capacity, and problem complexity. For example, empirically we find that disagreement goes down

when we increase the training set size, reduce the model's capacity (complexity), or reduce the inherent difficulty of the learning problem. Intuitively, the higher the disagreement rate, the higher the average error rate of the learner, where the average is taken over both test instances and training subsets. Therefore disagreement is a measure of the fitness of the learner to the learning task. However, as researchers have noted in relation to various measures of learner stability in general [Kut02], while robust learners (*i.e.*, algorithms with low prediction error) are stable, a stable learning algorithm does not necessarily have low prediction error. In the same vein, we show and explain that the disagreement measure provides only lower bounds on error. Still, our empirical results give evidence that disagreement can be a useful estimate in certain circumstances.

Since we require a source of unlabeled data—preferably a large source in order to accurately measure disagreement—we assume a *semi-supervised* setting where unlabeled data is relatively cheap and plentiful while labeled data is scarce or expensive. This scenario is often realistic, most notably for text classification. We focus on the binary classification setting and analyze 0/1 error.

In practice, cross validation—especially leave-one-out cross validation—often provides an accurate and reliable error estimate. In fact, under the usual assumption that training and test data both arise from the same distribution, $k$-fold cross validation provides an unbiased estimate of prediction error (for functions trained on $m(1 - 1/k)$ many instances, $m$ being the total number of labeled instances). However, in many situations, training data may actually arise from a different distribution than test data. One extreme example of this is *active learning*, where training samples are explicitly chosen to be maximally informative, using a process that is neither independent nor reflective of the test distribution. Even beyond active learning, in practice the process of gathering data and obtaining labels often may bias the training set, for example because some inputs are cheaper or easier to label, or are more readily available or obvious to the data collector, etc. In these cases, the error estimate obtained from cross validation may not yield an accurate measure of the prediction error of the learned function, and model selection based on cross validation may suffer. Empirically we find that in active learning settings, disagreement often provides a more accurate estimate of prediction error and is more useful as a guide for model selection.

Related to the problem of (average) error estimation is the problem of error variance estimation: both variance across test instances and variance across functions (i.e., training sets). Even if a learning algorithm exhibits relatively low average error, if it exhibits high variance, the algorithm may be undesirable depending on the end-user's risk tolerance. Variance is also useful for algorithm comparison, to determine whether observed error differences are statistically significant. For variance estimation, cross validation is on much less solid footing: in fact, Bengio and Grandvalet [BG03] recently proved an impossibility result showing that *no* method exists for producing an unbiased estimate of the variance of cross validation error in a pure supervised setting with labeled training data only. In this work, we show how disagreement relates to certain measures of variance. First, the disagreement on a particular instance provides an unbiased estimate of the variance of error on that instance. Second, disagreement provides an upper bound on the variance of prediction error (the type of variance useful for algorithm comparison).

The paper is organized as follows. In § 2 we formally define disagreement and prove how it lower-bounds prediction error and upper-bounds variance of prediction error. In § 3 we empirically explore how error estimates and model selection strategies that we devise based on disagreement compare against cross validation in standard (iid) learning settings and in active learning settings. In § 4 we discuss related work. We conclude in § 5.

## 2   Error, Variance, and Disagreement

Denote a set of input *instances* by $X$. Each instance $x \in X$ is a vector of feature attributes. Each instance has a unique true classification or *label* $y_x \in \{0, 1\}$, in general unknown to

the learner. Let $Z^* = \{(x, y_x)\}^m$ be a set of $m$ *labeled training* instances provided to the learner. The learner is an algorithm $\mathcal{A} : Z^* \rightarrow F$, that inputs labeled instances and output a function $f \in F$, where $F$ is the set of all functions (classifiers) that $\mathcal{A}$ may output (the hypothesis space). Each $f \in F$ is a function that maps instances $x$ to labels $\{0, 1\}$. The goal of the algorithm is to choose $f \in F$ to minimize 0/1 error (defined below) on future unlabeled *test* instances.

We assume the training set size is fixed at some $m > 0$, and we take expectations over one or both of two distributions: (1) the distribution $\mathcal{X}$ over instances in $X$, and (2) the distribution $\mathcal{F}$ induced over the functions $F$, when learner $\mathcal{A}$ is trained on training sets of size $m$ obtained by sampling from $\mathcal{X}$.

The 0/1 error $e_{x,f}$ of a given function $f$ on a given instance $x$ equals 1 if and only if the function incorrectly classifies the instances, and equals 0 otherwise; that is, $e_{x,f} = \mathbf{1}\{f(x) \neq y_x\}$. We define the *expected prediction error* $e$ of algorithm $\mathcal{A}$ as $e = E_{f,x}e_{f,x}$, where the expectation is taken over instances drawn from $\mathcal{X}$ ($x \sim \mathcal{X}$), and functions drawn from $\mathcal{F}$ ($f \sim \mathcal{F}$). The *variance of prediction error* $\sigma^2$ is useful for comparing different learners (*e.g.*, [BG03]). Let $e_f$ denote the 0/1 error of function $f$ (i.e., $e_f = E_x e_{x,f}$). Then $\sigma^2 = E_f((e_f - e)^2) = E_f(e_f^2) - e^2$.

Define the *disagreement* between two classifiers $f_1$ and $f_2$ on instance $x$ as $\mathbf{1}\{f_1(x) \neq f_2(x)\}$. The disagreement rate of learner $\mathcal{A}$ is then:

$$d = E_{x,f_1,f_2}\mathbf{1}\{f_1(x) \neq f_2(x)\}, \qquad (1)$$

where recall that the expectation is taken over $x \sim \mathcal{X}, f_1 \sim \mathcal{F}, f_2 \sim \mathcal{F}$ (with respect to traning sets of some fixed size $m$).

Let $d_x$ be the (expected) disagreement at $x$ when we sample functions from $\mathcal{F}$: $d_x = E_{f_1,f_2}\mathbf{1}\{f_1(x) \neq f_2(x)\}$. Similarly, let $e_x$ and $\sigma_x^2$ denote respectively the error and variance at $x$: $e_x = P(f(x) \neq y_x)) = E_f\mathbf{1}\{f(x) \neq y_x\} = E_f e_{f,x}$ and $\sigma_x^2 = VAR(e_f) = E_f[(\mathbf{1}\{f(x) \neq y_x\} - e_x)^2] = e_x(1 - e_x)$. (The last equality follow from the fact that $e_{f,x}$ is a Bernoulli/binary random variable.) Now, we can establish the connection between disagreement and variance of error (of the learner) at instance $x$:

$$
\begin{aligned}
d_x &= E_{f_1,f_2}\mathbf{1}\{(f_1(x) = y_x \text{ and } f_2(x) \neq y_x) \text{ or } (f_1(x) \neq y_x \text{ and} f_2(x) = y_x)\} \\
&= P(\mathbf{1}\{(f_1(x) = y_x \text{ and} f_2(x) \neq y_x) \text{ or } (f_1(x) \neq y_x \text{ and} f_2(x) = y_x)\} \\
&= 2P(f_1(x) = y_x \text{ and } f_2(x) \neq y_x) = 2e_x(1 - e_x) \Rightarrow
\end{aligned}
$$

$$\sigma_x^2 = d_x/2. \qquad (2)$$

The derivations follow from the fact that the expectation of a Bernoulli random variable is the same as its probability of being 1, and the two events above (the event $(f_1(x) = y_x$ and $f_2(x) \neq y_x)$ and the event $(f_1(x) = y_x$ and $f_2(x) \neq y_x)$ ) are mutually exclusive and have equal probability, and the two events $f_1(x) = y_x$ and $f_2(x) \neq y_x$ are conditionally independent (note that the two events are *conditioned on* $x$, and the two functions are picked independently of one another). Furthermore, $d = E_x E_{f_1,f_2}[\mathbf{1}\{f_1(x) \neq f_2(x)\}] = E_x d_x = 2E_x(\sigma_x^2) = 2E_x[e_x(1 - e_x)] = 2(e - E_x e_x^2)$, and therefore:

$$\frac{d}{2} = e - E_x e_x^2. \qquad (3)$$

## 2.1 Bounds on Variance via Disagreement

The variance of prediction error $\sigma^2$ can be used to test the significance of the difference in two learners' error rates. Bengio and Granvalet [BG03] show that there is no unbiased estimator of the variance of $k$-fold cross-validation in the supervised setting. We can see

from Equation 2 that having access to disagreement at a given instance $x$ (labeled or not) does yield the variance of error at that instance. Thus disagreement obtained via 2-fold training gives us an unbaised estimator of $\sigma_x^2$, the variance of prediction error at instance $x$, for functions trained on $m/2$ instances. (Note for unbiasedness, none of the functions should have been trained on the given instance.) Of course, to compare different algorithms on a given instance, one also needs the average error at that instance.

In terms of overall variance of prediction error $\sigma^2$ (where error is averaged across instances and variance taken across functions), there exist scenarios when $\sigma^2$ is 0 but $d$ is not (when errors of the different functions learned are the same but negatively correlated), and scenarios when $\sigma^2 = d/2 \neq 0$. In fact, disagreement yields an upper-bound:

**Theorem 1** $d \geq 2\sigma^2$.

**Proof (sketch).** We show that the result holds for any finite sampling of functions and instances: Consider the binary (0/1) matrix $M$ where the rows correspond to instances and the columns correspond to functions, and the entries are the binary-valued errors (entry $M_{i,j} = \mathbf{1}\{f_j(x_i) \neq y_{x_i}\}$). Thus the average error is the number of 1 entries when samplings of instances and functions are drawn from $\mathcal{X}$ and $\mathcal{F}$ respectively, and variances and disagreement can also be readily defined for the matrix. We show the inequality holds for any such $n \times n$ matrix for any $n$. This establishes the theorem (by using limiting arguments). Treat the 1 entries (matrix cells) as vertices in a graph, where an edge exists between two 1 entries if they share a column or a row. For a fixed number of 1 entries $N$ ($N \leq n^2$), we show the difference between disagreement and variance is minimized when the number of edges is maximized. We establish that configuration maximizing the number of edges occurs when all the 1 entries form a compact formation, that is, all the matrix entries in row i are filled before filling row i+1 with 1s. Finally, we show that for such a configuration minimzing the difference, the difference remains nonnegative. □

In typical small training sample size cases when the errors are nonzero and not entirely correlated (the patterns of 1s in the matrix is basically scattered) $d/2$ can be significantly larger than $\sigma^2$. With increasing training size, the functions learned tend to make the same errors and $d$ and $\sigma^2$ both approach 0.

## 2.2 Bounds on Error via Disagreement

From Jensen's inequality, we have that $E_x e_x^2 \geq (E_x e_x)^2 = e^2$, therefore using eq. 3, we conclude that $d/2 \leq e - e^2$. This implies that

$$\frac{1 - \sqrt{1 - 2d}}{2} \leq e \leq \frac{1 + \sqrt{1 - 2d}}{2}. \tag{4}$$

The upper bound derived is often not informative, as it is greater than $0.5$, and often we know the error is less than $0.5$. Let $e_l = \frac{1 - \sqrt{1 - 2d}}{2}$. We next discuss whether/when $e_l$ can be far from the actual error, and the related question of whether we can derive a good upperbound or just a good estimator on error using a measure based on disagreement.

When functions generated by the learner make correlated and frequent mistakes, $e_l$ can be far from true error. The extreme case of this is a learner that always outputs a constant function. In order to account for weak but stable learners, the error lower bound should be complemented with some measure that ensures that the learner is actually adapting (*i.e.*, doing its job!). We explore using the training (empirical) error for this purpose. Let $\tilde{e}$ denote the average training error of the algorithm: $\tilde{e} = E_f \tilde{e}_f = E_f \frac{1}{m} \sum_{x_i \in Z^*} \mathbf{1}\{f(x_i) \neq y_{x_i}\}$, where $Z^*$ is the training set that yielded $f$. Define $\hat{e} = \max(\tilde{e}, e_l)$. We explore $\hat{e}$ as a candidate criterion for model selection, which we compare against the cross-validation criterion in § 3.

Note that a learner *can* exhibit low disagreement and low training error, yet still have high prediction error. For example, the learner could memorize the training data and output a constant on all other instances. (Though when disagreement is exactly zero, the test error equals the training error.) A measure of self-disagreement within the labeled training set, defined by Lang et al. [LBRB02], in conjunction with the empirical training error does yield an upper bound. Still, we find empirically that, when using SVMs, naive Bayes, or logistic regression, disagreement on unlabeled data does not tend to wildly underestimate error, even though it's theoretically possible.

## 3 Experiments

We conducted experiments on the "20 Newsgroups" and Reuters-21578 test categorization datasets, and the Votes, Chess, Adult, and Optics datasets from the UCI collection [BKM98].[1] We chose two categorization tasks from the newsgroups sets: (1) identifying Baseball documents in a collection containing both Baseball and Hockey documents (2000 total documents), and (2) identifying alt.atheism documents from among the alt.atheism, soc.religion.christian, and talk.religion.misc collections (3000 documents). For the Reuters set, we chose documents belonging to one of the top 10 categories of the corpus (9410 documents), and we attempt to discriminate the "Earn" (3964) and "Acq" (2369) respectively from the remaining nine. These categories are large enough that 0/1 error remains a reasonable measure. We used the bow library for stemming and stop words, kept features up to 3-grams, and used l2-normalized frequency counts [McC96]. The Votes, Chess, Adult, and Optics datasets have respectively 435, 3197, 32561 and 1800 instances. These datasets give us some representation of the various types of learning problems. All our data set are in a nonnegative feature value representation. We used support vector machines with polynomial kernels available from the libsvm library [CL01] in all our experiments.[2] For the error estimation experiments, we used linear SVMs with a C value of 10. For the model selection experiments, we used polynomial degree as the model selection parameter.

### 3.1 Error Estimation

We first examine the use of disagreement for error estimation both in the standard setting where training and test samples are uniformly iid, and in an active learning scenario.

For each of several training set sizes for each data set, we computed average results and standard deviation across thirty trials. In each trial, we first generate a training set, sampled either uniformly iid or actively, then set aside 20% of remaining instances as the test set. Next, we partition the training set into equal halves, train an SVM on each half, and compute the disagreement rate between the two SVMs across the set of (unlabeled) data that has not been designated for the training or test set (80% of $total - m$ instances). We repeat this inner loop of partitioning, dual training, and disagreement computation thirty times and take averages.

We examined the utility of our disagreement bound (4) as an estimate of the true test error of the algorithm trained on the full data set ("trueE"). We also examined using the maximum of the training error ("trainE") and lower bound on error from our disagreement measure ("disE") as an estimate of trueE ("MaxDtE = max(trainE, disE)"). Note that disE and trainE are respectively unbiased empirical estimates of expected disagreement $d$ and expected training error $\tilde{e}$ of § 2 for the standard setting. Since our disagreement measure is actually a bound on *half error* (i.e., error averaged over training sets of size $m/2$), we also compare against two-fold cross-validation error ("2cvE"), and the true test error of the two functions obtained from training on the two halves ("1/2trueE").

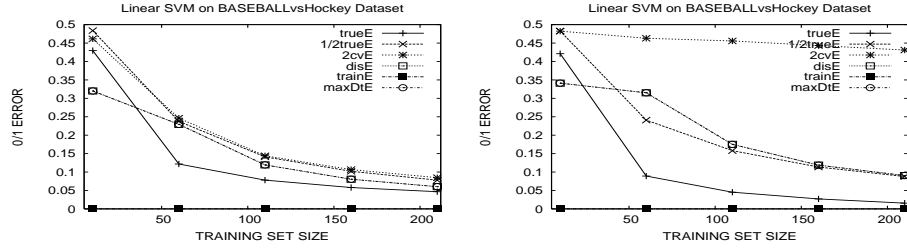

Figure 1: (a) Random training set. (b) Actively picked.

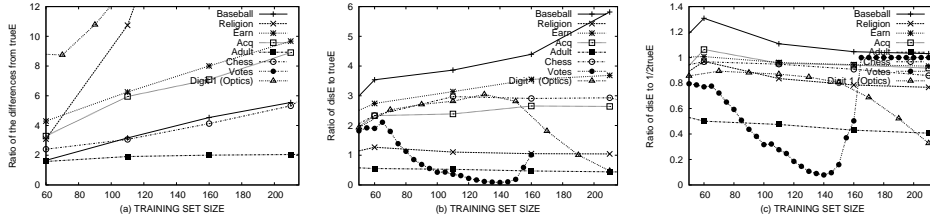

Figure 2: Plots of ratios when active learning: (a) $\frac{2cvE - trueE}{disE - trueE}$ (b) $\frac{disE}{trueE}$ (c) $\frac{disE}{1/2trueE}$.

In the standard scenario, when the training set is chosen uniformly at random from the corpus, leave-one-out cross validated error ("looE") is generally a very good estimate of trueE, while 2cvE is a good estimate for 1/2trueE. For all the data sets, as expected our error estimate maxDtE underestimates 1/2trueE. A representative example is shown in Figure 1(a).

In the active learning scenario, the training set is chosen in an attempt to maximize information, and the choice of each new instance depends on the set of previously chosen instances. Often this means that especially difficult instances are chosen (or at least instances whose labels are difficult to infer from the current training set). Thus cross validation naturally overestimates the difficulty of the learning task and so may greatly overestimate error. On the other hand, an approximate model of active learning is that the instances are *iid* sampled from a *hard* distribution. This ignores the sequential nature of active learning. Measuring disagreement on the easier test distribution via subsampling the training set may remain a good estimator of the actual test error.

We used linear SVMs as the basis for our active learning procedure. In each trial, we begin with random training set size of 10, and then grow the labeled set by using the uncertainty sampling technique. We computed the various error measures at regular intervals.[3] A representative plot of errors during active learning is given in Fig. 1(b). In all the datasets experimented with, we have observed the same pattern: the error estimate using disagreement provides a much better estimate of 1/2trueE and trueE than does 2cvE (Fig. 2a), and can be used as an indication of the error and the progress of active learning. Note that while we have not computed looE error in the error-estimation experiments, figure Fig. 1(b) indicates that 2cvE is not a good estimator of trueE at size $m/2$ either, and this has been the case in all our experiments. We have observed that disE estimates the 1/2trueE best (Fig. 2c). The estimation performance may degrade towards the end of active learning when the learner converges (disagreement approaches 0). However, we have observed that both 1/2trueE (obtained via subsampling) and disE tend to overestimate the actual error of the active learner even at half the training size (*e.g.*, Fig. 1(b)). This observation underlines the importance of taking the sequential nature of active learning into account.

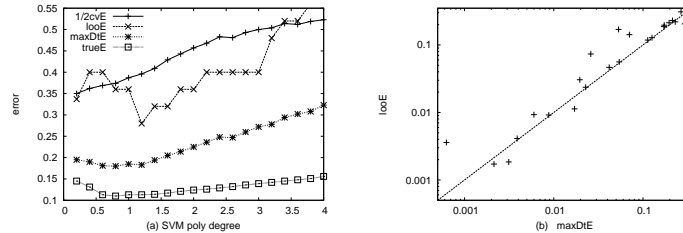

Figure 3: (a) An example were maxDtE performs particularly well as a model selection criteria, tracking the true error curve more closely than looE or 2cvE. (b) A summary of all experiments plotting looE versus maxDtE on a log-log scale: points above the diagonal indicate maxDtE outperforming looE.

## 3.2 Model Selection

We explore various criteria for selecting the expected best among twenty SVMs, each trained using a different polynomial degree kernel. For each data set, we manually identify an interval of polynomial degrees that seems to include the error minimum[4], then choose twenty degrees equally spaced within that interval. We compare our disagreement-based estimate maxDtE with the cross validation estimates looE and 2cvE as model selection criteria. In each trial, we identify the polynomial degree that is expected to be best according to each criteria, then train an SVM at that degree on the full training set. We compare trueE at the degree selected by each criteria against trueE at the actual optimal degree.

In the standard uniform iid scenario, though cross validation often does fail as a model selection criteria for *regression* problems, it seems that cross validation in general is hard to beat for classification problems [SS02]. We find that both looE and 2cvE modestly outperform maxDtE as model selection criteria, though maxDtE is often competitive. We are exploring using the maximum of cross validation and maxDtE as an alternative with preliminary evidence of a slight advantage over cross validation alone.

In an active learning setting, even though cross validation overestimates error, it is theoretically possible that cross validation would still function well to identify the best or near-best model. However, our experiments suggest that the performance of cross validation as a model selection criteria indeed degrades under active learning. In this situation, maxDtE serves as a consistently better model selection criteria. Figure 3(a) shows an example where maxDtE performs particularly well.

The active learning model selection experiments proceed as follows. For each data set, we use one run of active learning to identify 200 ordered and actively-picked instances. For each training size $m \in \{25, 50, 100, 200\}$, we run thirty experiments using a random shuffling of the size-$m$ prefix of the 200 actively-picked instances. In each trial and for each of the twenty polynomial degrees, we measure trueE and looE, then run an inner loop of thirty random partitionings and dual trainings to measure average $d$, expE, 2cvE, and 1/2trueE. Disagreements and errors are measured across the full test set ($total - m$ instances), so this is a *transductive* learning setting. Figure 3(b) summarizes the results. We observe that model selection based on disagreement often outperforms model selection based on cross-validation, and at times significantly so. Across 26 experiments, the win-loss-tie record of maxDtE versus 2cvE was 16-5-5, the record of maxDtE versus looE was 18-6-2, and the record of 2cvE versus looE was 15-9-2.

## 4  Related Work

Previous work has already shown that using various measures of stability on unlabeled data is useful for ensemble learning, model selection, and regularization, both in supervised and unsupervised learning [KV95, Sch97, SS02, BC03, LBRB02, LRBB04]. Metric-based methods for model selection are complementary to our approach in that they are desgined to prefer models/algorithms that behave similarly on the labeled and unlabeled data [Sch97, SS02, BC03], while disagreement is a measure of self-consistency on the same dataset (in this paper, unlabeled data only). Consequently, our method is also applicable to scenarios in which the test and training distributions are different. Lang et. al [LBRB02, LRBB04] also explore disagreement on unlabeled data, establishing robust model selection techniques based on disagreement for clustering. Theoretical work on algorithmic stability focuses on deriving generalization bounds given that the algorithm has certain inherent stability properties [KN02].

## 5  Conclusions and Future Work

Two advantages of co-validation over traditional techniques are: (1) disagreement can be measured to almost an arbitrary degree assuming unlabeled data is plentiful, and (2) disagreement is measured on unlabeled data drawn from the same distribution as test instances, the extreme case of which is in transductive learning where the unlabeled and test instances coincide. In this paper we derived bounds on certain measures of error and variance based on disagreement, then examined empirically when co-validation might be useful. We found co-validation particularly useful in active learning settings. Future goals include extending the theory to active learning, precision/recall, algorithm comparison (using variance), ensemble learning, and regression. We plan to compare semi-supervised and transductive learning, and consider procedures to generate fictitious unlabeled data.

## Footnotes

[1] Available from http://www.ics.uci.edu/ and http://www.daviddlewis.com/resources/testcollections/

[2] We observed similar results in error estimation using linear logistic regression and Naive Bayes learners in preliminary experiments.

[3]We could use a criterion based on disagreement for selective sampling, but we have not throughly explored this option.

[4]Although for fractional degress less than 1 the kernal matrix is not guaranteed to be positive semi-definite, we included such ranges whenever the range included the error minimum. Non-integral degress greater than 1 do not pose a problem as the feature values in all our problem representations are nonnegative.

### References

[BC03] Y. Bengio and N. Chapados. Extensions to metric-based model selection. *Journal of Machine Learning Research*, 2003.

[BG03] Y. Bengio and Y. Granvalet. No unbiased estimator of the variance of k-fold cross-validation. In *NIPS*, 2003.

[BKM98] C.L. Blake, E. Keogh, and C.J. Merz. UCI repository of machine learning databases, 1998.

[CL01] Chih-Chung Chang and Chih-Jen Lin. *LIBSVM: A library for support vector machines*, 2001. Software available at http://www.csie.ntu.edu.tw/~cjlin/libsvm.

[KN02] S. Kutin and P. Niyogi. Almost-everywhere algorithmic stability and generalization error. In *UAI*, 2002.

[Kut02] S. Kutin. *Algorithmic stability and ensemble-based learning*. PhD thesis, University of Chicago, 2002.

[KV95] A. Krogh and J. Vedelsby. Neural network ensembles, cross validation, and active learning. In *NIPS*, 1995.

[LBRB02] T. Lange, M. Braun, V. Roth, and J. Buhmann. Stability-based model selection. In *NIPS*, 2002.

[LRBB04] T. Lange, V. Roth, M. Braun, and J. Buhmann. Stability based validation of clustering algorithms. *Neural Computation*, 16, 2004.

[McC96] A. K. McCallum. Bow: A toolkit for statistical language modeling, text retrieval, classification and clustering. http://www.cs.cmu.edu/ mccallum/bow, 1996.

[Sch97] D. Schuurmans. A new metric-based approach to model selection. In *AAAI*, 1997.

[SS02] D. Schuurmans and F. Southey. Metric-based methods for adaptive model selection and regularization. *Machine Learning*, pages 51–84, 2002.
